# Improving Elevator Performance Using Reinforcement Learning

**Robert H. Crites**
Computer Science Department
University of Massachusetts
Amherst, MA 01003-4610
crites@cs.umass.edu

**Andrew G. Barto**
Computer Science Department
University of Massachusetts
Amherst, MA 01003-4610
barto@cs.umass.edu

## Abstract

This paper describes the application of reinforcement learning (RL) to the difficult real world problem of elevator dispatching. The elevator domain poses a combination of challenges not seen in most RL research to date. Elevator systems operate in continuous state spaces and in continuous time as discrete event dynamic systems. Their states are not fully observable and they are nonstationary due to changing passenger arrival rates. In addition, we use a team of RL agents, each of which is responsible for controlling one elevator car. The team receives a global reinforcement signal which appears noisy to each agent due to the effects of the actions of the other agents, the random nature of the arrivals and the incomplete observation of the state. In spite of these complications, we show results that in simulation surpass the best of the heuristic elevator control algorithms of which we are aware. These results demonstrate the power of RL on a very large scale stochastic dynamic optimization problem of practical utility.

## 1 INTRODUCTION

Recent algorithmic and theoretical advances in reinforcement learning (RL) have attracted widespread interest. RL algorithms have appeared that approximate dynamic programming (DP) on an incremental basis. Unlike traditional DP algorithms, these algorithms can perform with or without models of the system, and they can be used online as well as offline, focusing computation on areas of state space that are likely to be visited during actual control. On very large problems, they can provide computationally tractable ways of approximating DP. An example of this is Tesauro's TD–Gammon system (Tesauro, 1992; 1994; 1995), which used RL techniques to learn to play strong masters level backgammon. Even the

best human experts make poor teachers for this class of problems since they do not always know the best actions. Even if they did, the state space is so large that it would be difficult for experts to provide sufficient training data. RL algorithms are naturally suited to this class of problems, since they learn on the basis of their own experience. This paper describes the application of RL to elevator dispatching, another problem where classical DP is completely intractable. The elevator domain poses a number of difficulties that were not present in backgammon. In spite of these complications, we show results that surpass the best of the heuristic elevator control algorithms of which we are aware. The following sections describe the elevator dispatching domain, the RL algorithm and neural network architectures that were used, the results, and some conclusions.

## 2 THE ELEVATOR SYSTEM

The particular elevator system we examine is a simulated 10–story building with 4 elevator cars (Lewis, 1991; Bao et al, 1994). Passenger arrivals at each floor are assumed to be Poisson, with arrival rates that vary during the course of the day. Our simulations use a traffic profile (Bao et al, 1994) which dictates arrival rates for every 5–minute interval during a typical afternoon down–peak rush hour. Table 1 shows the mean number of passengers arriving at each floor (2–10) during each 5–minute interval who are headed for the lobby. In addition, there is inter–floor traffic which varies from 0% to 10% of the traffic to the lobby.

| Time | 00 | 05 | 10 | 15 | 20 | 25 | 30 | 35 | 40 | 45 | 50 | 55 |
|------|----|----|----|----|----|----|----|----|----|----|----|----|
| Rate | 1  | 2  | 4  | 4  | 18 | 12 | 8  | 7  | 18 | 5  | 3  | 2  |

Table 1: The Down–Peak Traffic Profile

The system dynamics are approximated by the following parameters:

- Floor time (the time to move one floor at the maximum speed): 1.45 secs.
- Stop time (the time needed to decelerate, open and close the doors, and accelerate again): 7.19 secs.
- Turn time (the time needed for a stopped car to change direction): 1 sec.
- Load time (the time for one passenger to enter or exit a car): random variable from a 20*th* order truncated Erlang distribution with a range from 0.6 to 6.0 secs and a mean of 1 sec.
- Car capacity: 20 passengers.

The state space is continuous because it includes the elapsed times since any hall calls were registered. Even if these real values are approximated as binary values, the size of the state space is still immense. Its components include $2^{18}$ possible combinations of the 18 hall call buttons (up and down buttons at each landing except the top and bottom), $2^{40}$ possible combinations of the 40 car buttons, and $18^4$ possible combinations of the positions and directions of the cars (rounding off to the nearest floor). Other parts of the state are not fully observable, for example, the desired destinations of the passengers waiting at each floor. Ignoring everything except the configuration of the hall and car call buttons and the approximate position and direction of the cars, we obtain an extremely conservative estimate of the size of a discrete approximation to the continuous state space:

$$2^{18} \cdot 2^{40} \cdot 18^4 \approx 10^{22} \text{ states.}$$

Each car has a small set of primitive actions. If it is stopped at a floor, it must either "move up" or "move down". If it is in motion between floors, it must either "stop at the next floor" or "continue past the next floor". Due to passenger expectations, there are two constraints on these actions: a car cannot pass a floor if a passenger wants to get off there and cannot turn until it has serviced all the car buttons in its present direction. We have added three additional action constraints in an attempt to build in some primitive prior knowledge: a car cannot stop at a floor unless someone wants to get on or off there, it cannot stop to pick up passengers at a floor if another car is already stopped there, and given a choice between moving up and down, it should prefer to move up (since the down–peak traffic tends to push the cars toward the bottom of the building). Because of this last constraint, the only real choices left to each car are the stop and continue actions. The actions of the elevator cars are executed asynchronously since they may take different amounts of time to complete.

The performance objectives of an elevator system can be defined in many ways. One possible objective is to minimize the average *wait* time, which is the time between the arrival of a passenger and his entry into a car. Another possible objective is to minimize the average *system* time, which is the sum of the wait time and the travel time. A third possible objective is to minimize the percentage of passengers that wait longer than some dissatisfaction threshold (usually 60 seconds). Another common objective is to minimize the sum of *squared* wait times. We chose this latter performance objective since it tends to keep the wait times low while also encouraging fair service.

## 3    THE ALGORITHM AND NETWORK ARCHITECTURE

Elevator systems can be modeled as *discrete event* systems, where significant events (such as passenger arrivals) occur at discrete times, but the amount of time between events is a real–valued variable. In such systems, the constant discount factor $\gamma$ used in most discrete–time reinforcement learning algorithms is inadequate. This problem can be approached using a variable discount factor that depends on the amount of time between events (Bradtke & Duff, 1995). In this case, returns are defined as integrals rather than as infinite sums, as follows:

$$\sum_{t=0}^{\infty} \gamma^t r_t \quad \text{becomes} \quad \int_0^{\infty} e^{-\beta \tau} r_\tau d\tau,$$

where $r_t$ is the immediate cost at discrete time $t$, $r_\tau$ is the instantaneous cost at continuous time $\tau$ (e.g., the sum of the squared wait times of all waiting passengers), and $\beta$ controls the rate of exponential decay.

Calculating reinforcements here poses a problem in that it seems to require knowledge of the waiting times of all waiting passengers. There are two ways of dealing with this problem. The simulator knows how long each passenger has been waiting. It could use this information to determine what could be called *omniscient* reinforcements. The other possibility is to use only information that would be available to a real system online. Such *online* reinforcements assume only that the waiting time of the first passenger in each queue is known (which is the elapsed button time). If the Poisson arrival rate $\lambda$ for each queue is estimated as the reciprocal of the last inter–button time for that queue, the Gamma distribution can be used to estimate the arrival times of subsequent passengers. The time until the $n^{th}$ subsequent arrival follows the Gamma distribution $\Gamma(n, \frac{1}{\lambda})$. For each queue, subsequent

arrivals will generate the following expected penalties during the first $b$ seconds after the hall button has been pressed:

$$\sum_{n=1}^{\infty} \int_0^b (\text{prob } n^{th} \text{ arrival occurs at time } \tau) \cdot (\text{penalty given arrival at time } \tau) \, d\tau$$

$$= \sum_{n=1}^{\infty} \int_0^b \frac{\lambda^n \tau^{n-1} e^{-\lambda\tau}}{(n-1)!} \int_0^{b-\tau} w^2 e^{-\beta(w+\tau)} dw \, d\tau = \int_0^b \int_0^{b-\tau} \lambda w^2 e^{-\beta(w+\tau)} dw \, d\tau.$$

This integral can be solved by parts to yield expected penalties. We found that using online reinforcements actually produced somewhat better results than using omniscient reinforcements, presumably because the algorithm was trying to learn average values anyway.

Because elevator system events occur randomly in continuous time, the branching factor is effectively infinite, which complicates the use of algorithms that require explicit lookahead. Therefore, we employed a team of discrete–event Q–learning agents, where each agent is responsible for controlling one elevator car. $Q(x, a)$ is defined as the expected infinite discounted return obtained by taking action $a$ in state $x$ and then following an optimal policy (Watkins, 1989). Because of the vast number of states, the Q–values are stored in feedforward neural networks. The networks receive some state information as input, and produce Q–value estimates as output. We have tested two architectures. In the parallel architecture, the agents share a single network, allowing them to learn from each other's experiences and forcing them to learn identical policies. In the fully decentralized architecture, the agents have their own networks, allowing them to specialize their control policies. In either case, none of the agents have explicit access to the actions of the other agents. Cooperation has to be learned indirectly via the global reinforcement signal. Each agent faces added stochasticity and nonstationarity because its environment contains other learning agents. Other work on team Q–learning is described in (Markey, 1994).

The algorithm calls for each car to select its actions probabilistically using the Boltzmann distribution over its Q–value estimates, where the temperature is lowered gradually during training. After every decision, error backpropagation is used to train the car's estimate of $Q(x, a)$ toward the following target output:

$$\int_{t_x}^{t_y} e^{-\beta(\tau - t_x)} r_\tau d\tau + e^{-\beta(t_y - t_x)} \min_b \hat{Q}(y, b),$$

where action $a$ is taken by the car from state $x$ at time $t_x$, the next decision by that car is required from state $y$ at time $t_y$, and $r_\tau$ and $\beta$ are defined as above. $e^{-\beta(t_y - t_x)}$ acts as a variable discount factor that depends on the amount of time between events. The learning rate parameter was set to 0.01 or 0.001 and $\beta$ was set to 0.01 in the experiments described in this paper.

After considerable experimentation, our best results were obtained using networks for pure down traffic with 47 input units, 20 hidden sigmoid units, and two linear output units (one for each action value). The input units are as follows:

- 18 units: Two units encode information about each of the nine down hall buttons. A real–valued unit encodes the elapsed time if the button has been pushed and a binary unit is on if the button has not been pushed.

- 16 units: Each of these units represents a possible location and direction for the car whose decision is required. Exactly one of these units will be on at any given time.
- 10 units: These units each represent one of the 10 floors where the other cars may be located. Each car has a "footprint" that depends on its direction and speed. For example, a stopped car causes activation only on the unit corresponding to its current floor, but a moving car causes activation on several units corresponding to the floors it is approaching, with the highest activations on the closest floors.
- 1 unit: This unit is on if the car whose decision is required is at the highest floor with a waiting passenger.
- 1 unit: This unit is on if the car whose decision is required is at the floor with the passenger that has been waiting for the longest amount of time.
- 1 unit: The bias unit is always on.

## 4    RESULTS

Since an optimal policy for the elevator dispatching problem is unknown, we measured the performance of our algorithm against other heuristic algorithms, including the best of which we were aware. The algorithms were: SECTOR, a sector–based algorithm similar to what is used in many actual elevator systems; DLB, Dynamic Load Balancing, attempts to equalize the load of all cars; HUFF, Highest Unanswered Floor First, gives priority to the highest floor with people waiting; LQF, Longest Queue First, gives priority to the queue with the person who has been waiting for the longest amount of time; FIM, Finite Intervisit Minimization, a receding horizon controller that searches the space of admissible car assignments to minimize a load function; ESA, Empty the System Algorithm, a receding horizon controller that searches for the fastest way to "empty the system" assuming no new passenger arrivals. ESA uses queue length information that would not be available in a real elevator system. ESA/nq is a version of ESA that uses arrival rate information to estimate the queue lengths. For more details, see (Bao et al, 1994). These receding horizon controllers are very sophisticated, but also very computationally intensive, such that they would be difficult to implement in real time. RLp and RLd denote the RL controllers, parallel and decentralized. The RL controllers were each trained on 60,000 hours of simulated elevator time, which took four days on a 100 MIPS workstation. The results are averaged over 30 hours of simulated elevator time. Table 2 shows the results for the traffic profile with down traffic only.

| Algorithm | AvgWait | SquaredWait | SystemTime | Percent>60 secs |
|---|---|---|---|---|
| SECTOR | 21.4 | 674 | 47.7 | 1.12 |
| DLB | 19.4 | 658 | 53.2 | 2.74 |
| BASIC HUFF | 19.9 | 580 | 47.2 | 0.76 |
| LQF | 19.1 | 534 | 46.6 | 0.89 |
| HUFF | 16.8 | 396 | 48.6 | 0.16 |
| FIM | 16.0 | 359 | 47.9 | 0.11 |
| ESA/nq | 15.8 | 358 | 47.7 | 0.12 |
| ESA | 15.1 | 338 | 47.1 | 0.25 |
| RLp | 14.8 | 320 | 41.8 | 0.09 |
| RLd | 14.7 | 313 | 41.7 | 0.07 |

Table 2: Results for Down–Peak Profile with Down Traffic Only

Table 3 shows the results for the down–peak traffic profile with up and down traffic, including an average of 2 up passengers per minute at the lobby. The algorithm was trained on down–only traffic, yet it generalizes well when up traffic is added and upward moving cars are forced to stop for any upward hall calls.

| Algorithm | AvgWait | SquaredWait | SystemTime | Percent>60 secs |
|---|---|---|---|---|
| SECTOR | 27.3 | 1252 | 54.8 | 9.24 |
| DLB | 21.7 | 826 | 54.4 | 4.74 |
| BASIC HUFF | 22.0 | 756 | 51.1 | 3.46 |
| LQF | 21.9 | 732 | 50.7 | 2.87 |
| HUFF | 19.6 | 608 | 50.5 | 1.99 |
| ESA | 18.0 | 524 | 50.0 | 1.56 |
| FIM | 17.9 | 476 | 48.9 | 0.50 |
| RLp | 16.9 | 476 | 42.7 | 1.53 |
| RLd | 16.9 | 468 | 42.7 | 1.40 |

Table 3: Results for Down–Peak Profile with Up and Down Traffic

Table 4 shows the results for the down–peak traffic profile with up and down traffic, including an average of 4 up passengers per minute at the lobby. This time there is twice as much up traffic, and the RL agents generalize extremely well to this new situation.

| Algorithm | AvgWait | SquaredWait | SystemTime | Percent>60 secs |
|---|---|---|---|---|
| SECTOR | 30.3 | 1643 | 59.5 | 13.50 |
| HUFF | 22.8 | 884 | 55.3 | 5.10 |
| DLB | 22.6 | 880 | 55.8 | 5.18 |
| LQF | 23.5 | 877 | 53.5 | 4.92 |
| BASIC HUFF | 23.2 | 875 | 54.7 | 4.94 |
| FIM | 20.8 | 685 | 53.4 | 3.10 |
| ESA | 20.1 | 667 | 52.3 | 3.12 |
| RLd | 18.8 | 593 | 45.4 | 2.40 |
| RLp | 18.6 | 585 | 45.7 | 2.49 |

Table 4: Results for Down–Peak Profile with Twice as Much Up Traffic

One can see that both the RL systems achieved very good performance, most notably as measured by system time (the sum of the wait and travel time), a measure that was not directly being minimized. Surprisingly, the decentralized RL system was able to achieve as good a level of performance as the parallel RL system. Better performance with nonstationary traffic profiles may be obtainable by providing the agents with information about the current traffic context as part of their input representation. We expect that an additional advantage of RL over heuristic controllers may be in buildings with less homogeneous arrival rates at each floor, where RL can adapt to idiosyncracies in their individual traffic patterns.

## 5   CONCLUSIONS

These results demonstrate the utility of RL on a very large scale dynamic optimization problem. By focusing computation onto the states visited during simulated trajectories, RL avoids the need of conventional DP algorithms to exhaustively

sweep the state set. By storing information in artificial neural networks, it avoids the need to maintain large lookup tables. To achieve the above results, each RL system experienced 60,000 hours of simulated elevator time, which took four days of computer time on a 100 MIPS processor. Although this is a considerable amount of computation, it is negligible compared to what any conventional DP algorithm would require. The results also suggest that approaches to decentralized control using RL have considerable promise. Future research on the elevator dispatching problem will investigate other traffic profiles and further explore the parallel and decentralized RL architectures.

## Acknowledgements

We thank John McNulty, Christos Cassandras, Asif Gandhi, Dave Pepyne, Kevin Markey, Victor Lesser, Rod Grupen, Rich Sutton, Steve Bradtke, and the ANW group for assistance with the simulator and for helpful discussions. This research was supported by the Air Force Office of Scientific Research under grant F49620–93–1–0269.

## References

G. Bao, C. G. Cassandras, T. E. Djaferis, A. D. Gandhi, and D. P. Looze. (1994) *Elevator Dispatchers for Down Peak Traffic*. Technical Report, ECE Department, University of Massachusetts, Amherst, MA.

S. J. Bradtke and M. O. Duff. (1995) Reinforcement Learning Methods for Continuous–Time Markov Decision Problems. In: G. Tesauro, D. S. Touretzky and T. K. Leen, eds., *Advances in Neural Information Processing Systems 7*, MIT Press, Cambridge, MA.

J. Lewis. (1991) *A Dynamic Load Balancing Approach to the Control of Multiserver Polling Systems with Applications to Elevator System Dispatching*. PhD thesis, University of Massachusetts, Amherst, MA.

K. L. Markey. (1994) Efficient Learning of Multiple Degree–of–Freedom Control Problems with Quasi-independent Q-agents. In: M. C. Mozer, P. Smolensky, D. S. Touretzky, J. L. Elman and A. S. Weigend, eds., *Proceedings of the 1993 Connectionist Models Summer School*. Erlbaum Associates, Hillsdale, NJ.

G. Tesauro. (1992) Practical Issues in Temporal Difference Learning. *Machine Learning* 8:257–277.

G. Tesauro. (1994) TD–Gammon, a Self–Teaching Backgammon Program, Achieves Master-Level Play. *Neural Computation* 6:215–219.

G. Tesauro. (1995) Temporal Difference Learning and TD–Gammon. *Communications of the ACM* 38:58–68.

C. J. C. H. Watkins. (1989) *Learning from Delayed Rewards*. PhD thesis, Cambridge University.